# Real-Time Monitoring of Complex Industrial Processes with Particle Filters

**Rubén Morales-Menéndez**[*]
Dept. of Mechatronics and Automation
ITESM campus Monterrey
Monterrey, NL México
rmm@itesm.mx

**Nando de Freitas and David Poole**
Dept. of Computer Science
University of British Columbia
Vancouver, BC, V6T 1Z4, Canada
{nando,poole}@cs.ubc.ca

## Abstract

This paper discusses the application of particle filtering algorithms to fault diagnosis in complex industrial processes. We consider two ubiquitous processes: an industrial dryer and a level tank. For these applications, we compared three particle filtering variants: standard particle filtering, Rao-Blackwellised particle filtering and a version of Rao-Blackwellised particle filtering that does one-step look-ahead to select good sampling regions. We show that the overhead of the extra processing per particle of the more sophisticated methods is more than compensated by the decrease in error and variance.

## 1 Introduction

Real-time monitoring is important in many areas such as robot navigation or diagnosis of complex systems [1, 2]. This paper considers online monitoring of complex industrial processes. The processes have a number of discrete states, corresponding to different combinations of faults or regions of qualitatively different dynamics. The dynamics can be very different based on the discrete states. Even if there are very few discrete states, exact monitoring is computationally unfeasible as the state of the system depends on the history of the discrete states. However there is a need to monitor these systems in real time to determine what faults could have occurred.

This paper investigates the feasibility of using particle filtering (PF) for online monitoring. It also proposes some powerful variants of PF. These variants involve doing more computation per particle for each time step. We wanted to investigate whether we could do real-time monitoring and whether the extra cost of the more sophisticated methods was worthwhile in these real-world domains.

## 2 Classical approaches to fault diagnosis in dynamic systems

Most existing model-based fault diagnosis methods use a technique called *analytical redundancy* [3]. Real measurements of a process variable are compared to analytically calculated

---

[*] Visiting Scholar (2000-2003) at The University of British Columbia.

values. The resulting differences, named *residuals*, are indicative of faults in the process. Many of these methods rely on simplifications and heuristics [4, 5, 6, 7]. Here, we propose a principled probabilistic approach to this problem.

## 3  Processes monitored

We analyzed two industrial processes: an *industrial dryer* and a *level-tank*. In each of these, we physically inserted a sequence of faults into the system and made appropriate measurements. The nonlinear models that we used in the stochastic simulation were obtained through open-loop step responses for each discrete state [8]. The parametric identification procedure was guided by the minimum squares error algorithm [9] and validated with the "Control Station" software [10]. The discrete-time state space representation was obtained by a standard procedure in control engineering [8].

### 3.1  Industrial dryer

An industrial dryer is a thermal process that converts electricity to heat. As shown in Figure 1, we measure the temperature of the output air-flow.

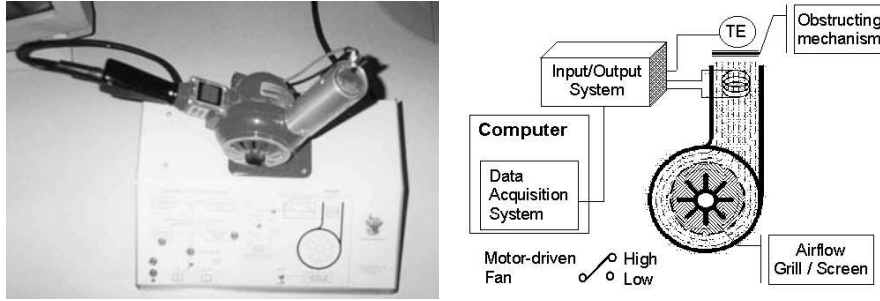

Figure 1: Industrial dryer.

Normal operation corresponds to low fan speed, open air-flow grill and clean temperature sensor (we denote this discrete state $z_t = 1$). We induced 3 types of fault: $z_t = 2$ faulty fan, $z_t = 3$ faulty grill (the grill is closed), and $z_t = 4$ faulty fan and grill.

### 3.2  Level tank

Many continuous industrial processes need to control the amount of accumulated material using level measurement, such as evaporators, distillation columns or boilers. We worked with a level-tank system that exhibits the dynamic behaviour of these important processes, see Figure 2. A by-pass pipe and two manual valves ($V_1$ and $V_2$) where used to induce typical faulty states.

## 4  Mathematical model

We adopted the following jump Markov linear Gaussian model:

$$\begin{aligned}
z_t &\sim & P(z_t|z_{t-1}) \\
x_t &=& A(z_t)x_{t-1} + B(z_t)w_t + F(z_t)u_t \\
y_t &=& C(z_t)x_t + D(z_t)v_t + G(z_t)u_t,
\end{aligned}$$

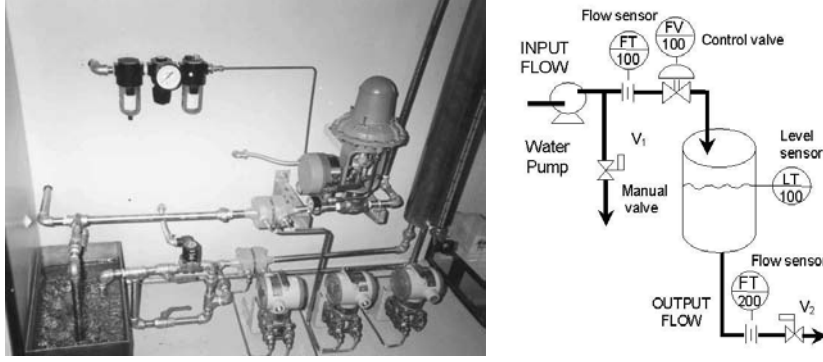

Figure 2: Level-Tank

where $y_t \in \mathbb{R}^{n_y}$ denotes the measurements, $x_t \in \mathbb{R}^{n_x}$ denotes the unknown continuous states, $u_t \in \mathcal{U}$ is a known control signal, $z_t \in \{1, \ldots, n_z\}$ denotes the unknown discrete states (normal operations and faulty conditions). The noise processes are *i.i.d* Gaussian: $w_t \sim \mathcal{N}(0, I)$ and $v_t \sim \mathcal{N}(0, I)$. The parameters $(A, B, C, D, F, G, P(z_t|z_{t-1}))$ are identified matrices with $D(z_t)D(z_t)^\mathsf{T} > 0$ for any $z_t$. The initial states are $x_0 \sim \mathcal{N}(\mu_0, \Sigma_0)$ and $z_0 \sim P(z_0)$. The important thing to notice is that for each realization of $z_t$, we have a single linear-Gaussian model. If we knew $z_t$, we could solve for $x_t$ exactly using the Kalman filter algorithm.

The aim of the analysis is to compute the marginal posterior distribution[1] of the discrete states $P(z_{0:t}|y_{1:t})$. This distribution can be derived from the posterior distribution $P(dx_{0:t}, z_{0:t}|y_{1:t})$ by standard marginalisation. The posterior density satisfies the following recursion

$$p\left(x_{0:t}, z_{0:t}|y_{1:t}\right) \;\; = \;\; p\left(x_{0:t-1}, z_{0:t-1}|y_{1:t-1}\right) \frac{p\left(y_t|x_t, z_t\right) p\left(x_t, z_t|x_{t-1}, z_{t-1}\right)}{p\left(y_t|y_{1:t-1}\right)}. \quad (1)$$

This recursion involves intractable integrals. One, therefore, has to resort to some form of numerical approximation scheme.

## 5   Particle filtering

In the PF setting, we use a weighted set of samples (particles) $\{(x_{0:t}^{(i)}, z_{0:t}^{(i)}), w_t^{(i)}\}_{i=1}^N$ to approximate the posterior with the following point-mass distribution

$$\widehat{P}_N(dx_{0:t}, z_{0:t}|y_{1:t}) = \sum_{i=1}^N w_t^{(i)} \delta_{x_{0:t}^{(i)}, z_{0:t}^{(i)}}(dx_{0:t}, z_{0:t}),$$

where $\delta_{x_{0:t}^{(i)}, z_{0:t}^{(i)}}(dx_{0:t}, z_{0:t})$ denotes the Dirac-delta function. Given $N$ particles $\{x_{0:t-1}^{(i)}, z_{0:t-1}^{(i)}\}_{i=1}^N$ at time $t-1$, approximately distributed according to

$P(dx_{0:t-1}^{(i)}, z_{0:t-1}^{(i)}|y_{1:t-1})$, PF enables us to compute $N$ particles $\{x_{0:t}^{(i)}, z_{0:t}^{(i)}\}_{i=1}^{N}$ approximately distributed according to $P(dx_{0:t}^{(i)}, z_{0:t}^{(i)}|y_{1:t})$, at time $t$. Since we cannot sample from the posterior directly, the PF update is accomplished by introducing an appropriate importance proposal distribution $Q(dx_{0:t}, z_{0:t})$ from which we can obtain samples. The basic algorithm, Figure (3), consists of two steps: sequential importance sampling and selection (see [11] for a detailed derivation). This algorithm uses the transition priors as proposal distributions; $Q(x_{0:t}, z_{0:t}|y_{1:t}) = P(x_t|x_{t-1}, z_t)P(z_t|z_{t-1})$. For the selection step, we used a state-of-the-art minimum variance resampling algorithm [12].

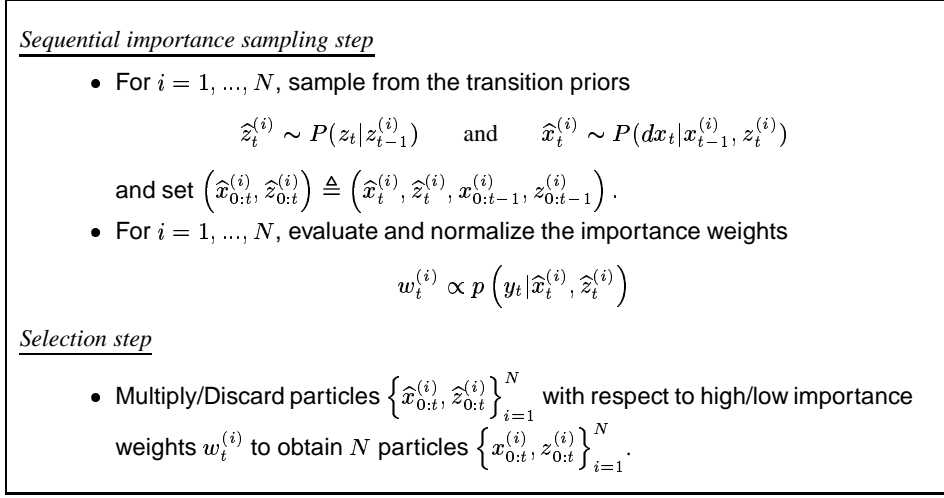

Figure 3: PF algorithm at time $t$.

## 6   Improved Rao-Blackwellised particle filtering

By considering the factorisation $p\left(x_{0:t}, z_{0:t}|\, y_{1:t}\right) = p\left(x_{0:t}|\, y_{1:t}, z_{0:t}\right)p\left(z_{0:t}|\, y_{1:t}\right)$, it is possible to design more efficient algorithms. *The density $p\left(x_{0:t}|\, y_{1:t}, z_{0:t}\right)$ is Gaussian and can be computed analytically if we know the marginal posterior density $p\left(z_{0:t}|\, y_{1:t}\right)$.* This density satisfies the alternative recursion

$$p\left(z_{0:t}|y_{1:t}\right) = p\left(z_{0:t-1}|y_{1:t-1}\right)\frac{p\left(y_t|y_{1:t-1}, z_{0:t}\right)p\left(z_t|z_{t-1}\right)}{p\left(y_t|y_{1:t-1}\right)} \qquad (2)$$

If equation (1) does not admit a closed-form expression, then equation (2) does not admit one either and sampling-based methods are still required. (Also note that the term $p\left(y_t|y_{1:t-1}, z_{0:t}\right)$ in equation (2) does not simplify to $p\left(y_t|z_t\right)$ because there is a dependency on past values through $x_{0:t}$.) Now assuming that we can use a weighted set of samples $\{z_{0:t}^{(i)}, w_t^{(i)}\}_{i=1}^{N}$ to represent the marginal posterior distribution

$$\widehat{P}_N(z_{0:t}|y_{1:t}) = \sum_{i=1}^{N} w_t^{(i)}\delta_{z_{0:t}^{(i)}}(z_{0:t}),$$

the marginal density of $x_{0:t}$ is a Gaussian mixture

$$\widehat{p}_N(x_{0:t}|y_{1:t}) = \int p(x_{0:t}|z_{0:t}, y_{1:t})dP(z_{0:t}|y_{1:t}) = \sum_{i=1}^{N} w_t^{(i)}p(x_{0:t}|y_{1:t}, z_{0:t}^{(i)})$$

that can be computed efficiently with a stochastic bank of Kalman filters. That is, we use PF to estimate the distribution of $z_t$ and exact computations (Kalman filter) to estimate the mean and variance of $z_t$. In particular, we sample $z_t^{(i)}$ and then propagate the mean $\mu_t^{(i)}$ and covariance $\Sigma_t^{(i)}$ of $x_t$ with a Kalman filter:

$$
\begin{aligned}
\mu_{t|t-1}^{(i)} &= A(z_t^{(i)})\mu_{t-1}^{(i)} + F(z_t^{(i)})u_t \\
\Sigma_{t|t-1}^{(i)} &= A(z_t^{(i)})\Sigma_{t-1}^{(i)}A(z_t^{(i)})^{\mathrm{T}} + B(z_t^{(i)})B(z_t^{(i)})^{\mathrm{T}} \\
S_t^{(i)} &= C(z_t^{(i)})\Sigma_{t|t-1}^{(i)}C(z_t^{(i)})^{\mathrm{T}} + D(z_t^{(i)})D(z_t^{(i)})^{\mathrm{T}} \\
y_{t|t-1}^{(i)} &= C(z_t^{(i)})\mu_{t|t-1}^{(i)} + G(z_t^{(i)})u_t \\
\mu_t^{(i)} &= \mu_{t|t-1}^{(i)} + \Sigma_{t|t-1}^{(i)}C(z_t^{(i)})^{\mathrm{T}}S_t^{-1(i)}(y_t - y_{t|t-1}^{(i)}) \\
\Sigma_t^{(i)} &= \Sigma_{t|t-1}^{(i)} - \Sigma_{t|t-1}^{(i)}C(z_t^{(i)})^{\mathrm{T}}S_t^{-1(i)}C(z_t^{(i)})\Sigma_{t|t-1}^{(i)},
\end{aligned}
$$

where $\mu_{t|t-1} \triangleq \mathbb{E}(x_t|y_{1:t-1})$, $\mu_t \triangleq \mathbb{E}(x_t|y_{1:t})$, $y_{t|t-1} \triangleq \mathbb{E}(y_t|y_{1:t-1})$, $\Sigma_{t|t-1} \triangleq cov(x_t|y_{1:t-1})$, $\Sigma_t \triangleq cov(x_t|y_{1:t})$ and $S_t \triangleq cov(y_t|y_{1:t-1})$.

This is the basis of the RBPF algorithm that was adopted in [13]. Here, we introduce an extra improvement. Let us expand the expression for the importance weights:

$$
w_t = \frac{p(z_{0:t}|y_{1:t})}{q(z_{0:t}|y_{1:t})} = \frac{p(z_{0:t-1}|y_{1:t})}{p(z_{0:t-1}|y_{1:t-1})} \frac{p(z_t|z_{0:t-1},y_{1:t})}{q(z_t|z_{0:t-1},y_{1:t})} \tag{3}
$$

$$
\propto \frac{p(y_t|y_{1:t-1},z_{0:t})\, p(z_t|z_{0:t-1},y_{1:t-1})}{q(z_t|z_{0:t-1},y_{1:t})}. \tag{4}
$$

The proposal choice, $q(z_{0:t}|y_{1:t}) = q(z_t|z_{0:t-1},y_{1:t})p(z_{0:t-1}|y_{1:t-1})$, states that we are not sampling past trajectories. Sampling past trajectories requires solving an intractable integral [14].

We could use the transition prior as proposal distribution: $q(z_t|z_{0:t-1},y_{1:t}) = p(z_t|z_{0:t-1},y_{1:t-1}) = p(z_t|z_{t-1})$. Then, according to equation (4), the importance weights simplify to the predictive density

$$
w_t \propto p(y_t|y_{1:t-1},z_{0:t}) = \mathcal{N}(y_t; y_{t|t-1}, S_t). \tag{5}
$$

However, we can do better by noticing that according to equation (3), the optimal proposal distribution corresponds to the choice $q(z_t|z_{0:t-1},y_{1:t}) = p(z_t|z_{0:t-1},y_{1:t})$. This distribution satisfies Bayes rule:

$$
p(z_t|z_{0:t-1},y_{1:t}) = \frac{p(y_t|y_{1:t-1},z_{0:t})\, p(z_t|z_{0:t-1},y_{1:t-1})}{p(y_t|y_{1:t-1},z_{0:t-1})} \tag{6}
$$

and, hence, the importance weights simplify to

$$
w_t \propto p(y_t|y_{1:t-1},z_{0:t-1}) = \sum_{z_t=1}^{n_z} p(y_t|y_{1:t-1},z_{0:t-1},z_t)\, p(z_t|z_{t-1}) \tag{7}
$$

*When the number of discrete states is small, say 10 or 100, we can compute the distributions in equations (6) and (7) analytically.* In addition to Rao-Blackwellisation, this leads to substantial improvements over standard particle filters. Yet, a further improvement can still be attained.

Even when using the optimal importance distribution, there is a discrepancy arising from the ratio $p(z_{0:t-1}|y_{1:t})/p(z_{0:t-1}|y_{1:t-1})$ in equation (3). This discrepancy is what causes the well known problem of sample impoverishment in all particle filters [11, 15]. To circumvent it to a significant extent, we note that the importance weights do not depend on

$z_t$ (we are marginalising over this variable). *It is therefore possible to select particles before the sampling step. That is, one chooses the fittest particles at time $t - 1$ using the information at time $t$.* This observation leads to an efficient algorithm (*look-ahead* RBPF), whose pseudocode is shown in Figure 4. Note that for standard PF, Figure 3, the importance weights depend on the sample $z_t^{(i)}$, thus not permitting selection before sampling. Selecting particles before sampling results in a richer sample set at the end of each time step.

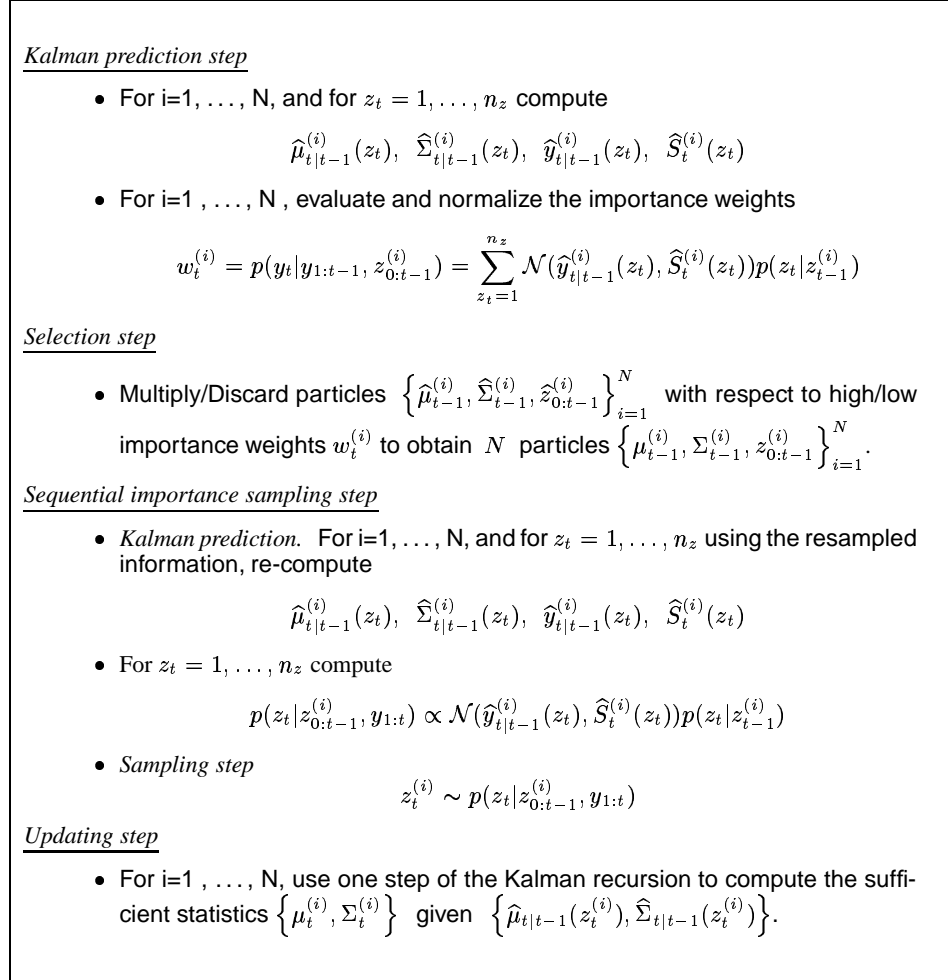

Figure 4: look-ahead RBPF algorithm at time $t$. The algorithm uses an optimal proposal distribution. It also selects particles from time $t - 1$ using the information at time $t$.

## 7   Results

The results are shown in Figures 5 and 6. The left graphs show error detection versus computing time per time-step (the signal sampling time was 1 second). The right graphs show the error detection versus number of particles. The error detection represents how

many discrete states were not identified properly, and was calculated for 25 independent runs (1,000 time steps each). The graphs show that look-ahead RBPF works significantly better (low error rate and very low variance). This is essential for real-time operation with low error rates.

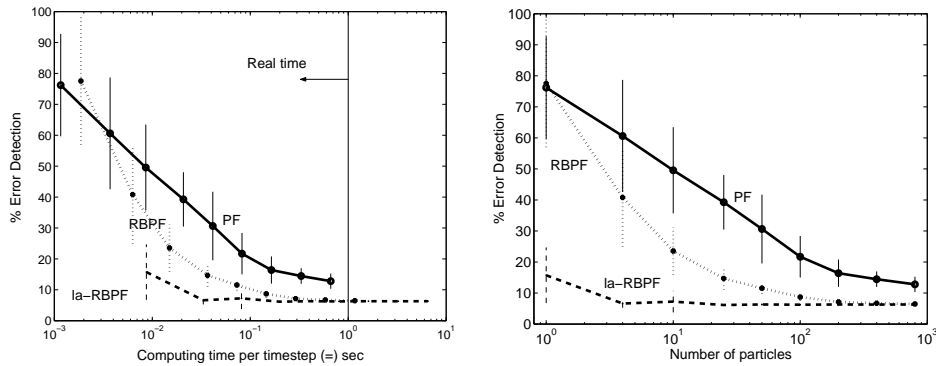

Figure 5: Industrial dryer: error detection vs computing time and number of particles.

The graphs also show that even for 1 particle, look-ahead RBPF is able to track the discrete state. The reason for this is that the sensors are very accurate (variance=0.01). Consequently, the distributions are very peaked and we are simply tracking the mode. Note that look-ahead RBPF is the only filter that uses the most recent information in the proposal distribution. Since the measurements are very accurate, it finds the mode easily. We repeated the level-tank experiments with noisier sensors (variance=0.08) and obtained the results shown in Figure 7. Noisier sensors, as expected, reduce the accuracy of look-ahead RBPF with a small number of particles. However, it is still possible to achieve low error rates in real-time. Since modern industrial and robotic sensors tend to be very accurate, we conclude that look-ahead RBPF has great potential.

### Acknowledgments

Ruben Morales-Menéndez was partly supported by the Government of Canada (ICCS) and UBC CS department. David Poole and Nando de Freitas are supported by NSERC

## Footnotes

[1]NOTATION: For a generic vector $\theta$, we adopt the notation $\theta_{1:t} \triangleq (\theta_1, \theta_2, \ldots, \theta_t)'$ to denote all the entries of this vector at time $t$. For simplicity, we use $\theta_t$ to denote both the random variable and its realisation. Consequently, we express continuous probability distributions using $P(d\theta_t)$ instead of $\Pr(\theta_t \in d\theta_t)$ and discrete distributions using $P(\theta_t)$ instead of $\Pr(\theta_t = \theta_t)$. If these distributions admit densities with respect to an underlying measure $\mu$ (counting or Lebesgue), we denote these densities by $p(\theta_t)$. For example, when considering the space $\mathbb{R}^n$, we will use the Lebesgue measure, $\mu = d\theta_t$, so that $P(d\theta_t) = p(\theta_t) d\theta_t$.

### References

[1] J Chen and J Howell. A self-validating control system based approach to plant fault detection and diagnosis. *Computers and Chemical Engineering*, 25:337–358, 2001.

[2] S Thrun, J Langford, and V Verma. Risk sensitive particle filters. In S Becker T. G Dietterich and Z Ghahramani, editors, *Advances in Neural Information Processing Systems 14*, Cambridge, MA, 2002. MIT Press.

[3] J Gertler. *Fault detection and diagnosis in engineering systems.* Marcel Dekker, Inc., 1998.

[4] P Frank and X Ding. Survey of robust residual generation and evaluation methods in observer-based fault detection systems. *J. Proc. Cont*, 7(6):403–424, 1997.

[5] P Frank, E Alcorta Garcia, and B.Koppen-Seliger. Modelling for fault detection and isolation versus modelling for control. *Mathematics and computers in simulation*, 53:259—271, 2000.

[6] P Frank. Fault diagnosis in dynamic systems using analytical and knowledge-based redundancy – a survey and some new results. *Automatica*, 26:459–474, 1990.

[7] R Isermann. Supervision, fault-detection and fault-diagnosis methods - an introduction. *Control engineering practice*, 5(5):639–652, 1997.

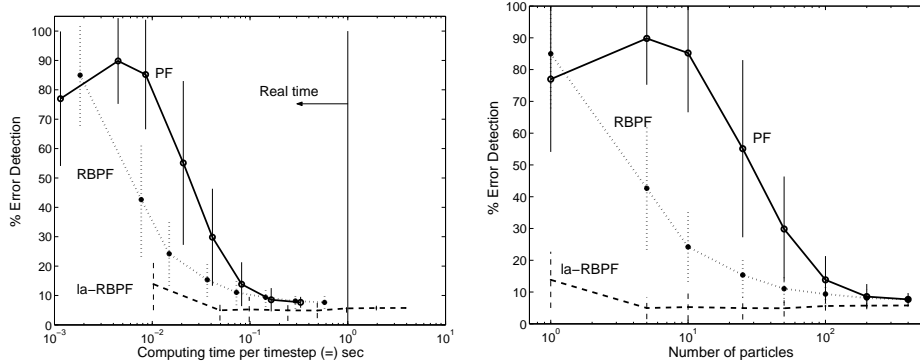

Figure 6: Level-tank (accurate sensors): error detection vs computing time and number of particles.

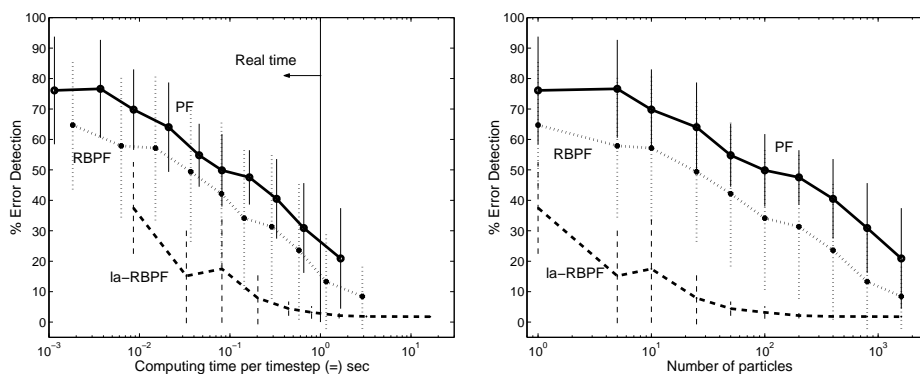

Figure 7: Level-tank (noisy sensors): error detection vs computing time and number of particles.

[8] K Ogata. *Discrete-Time Control Systems*. Prentice Hall, second edition, 1995.

[9] L Ljung. *System Identification: Theory for the User*. Prentice-Hall, 1987.

[10] D Cooper. *Control Station*. University of Connecticut, third edition, 2001.

[11] A Doucet, N de Freitas, and N J Gordon, editors. *Sequential Monte Carlo Methods in Practice*. Springer-Verlag, 2001.

[12] G Kitagawa. Monte Carlo filter and smoother for non-Gaussian nonlinear state space models. *Journal of Computational and Graphical Statistics*, 5:1–25, 1996.

[13] N de Freitas. Rao-Blackwellised particle filtering for fault diagnosis. In *IEEE aerospace conference*, 2001.

[14] C Andrieu, A Doucet, and E Punskaya. Sequential Monte Carlo methods for optimal filtering. In A Doucet, N de Freitas, and N J Gordon, editors, *Sequential Monte Carlo Methods in Practice*. Springer-Verlag, 2001.

[15] M Pitt and N Shephard. Filtering via simulation: auxiliary particle filters. *Journal of the American statistical association*, 94(446):590–599, 1999.
